# A robust inlier identification algorithm for point cloud registration via $\ell_0$-minimization

**Yinuo Jiang**[1]* **Xiuchuan Tang**[2]* **Cheng Cheng**[1] **Ye Yuan**[1]†

[1]School of Artificial Intelligence and Automation, Huazhong University of Science and Technology, China
[2]Department of Automation, Tsinghua University, China
`{jiangyinuo,c_cheng,yye}@hust.edu.cn; tangxiuchuan@mail.tsinghua.edu.cn`
`https://github.com/HAIRLAB/inlier-identification-via-l0`

## Abstract

Correspondences in point cloud registration are prone to outliers, significantly reducing registration accuracy and highlighting the need for precise inlier identification. In this paper, we propose a robust inlier identification algorithm for point cloud registration by reformulating the conventional registration problem as an alignment error $\ell_0$-minimization problem. The $\ell_0$-minimization problem is formulated for each local set, where those local sets are built on a compatibility graph of input correspondences. To resolve the $\ell_0$-minimization, we develop a novel two-stage decoupling strategy, which first decouples the alignment error into a rotation fitting error and a translation fitting error. Second, null-space matrices are employed to decouple inlier identification from the estimation of rotation and translation respectively, thereby applying Bayes Theorem to $\ell_0$-minimization problems and solving for fitting errors. Correspondences with the smallest errors are identified as inliers to generate a transformation hypothesis for each local set. The best hypothesis is selected to perform registration. We demonstrate that the proposed inlier identification algorithm is robust under high outlier ratios and noise through experiments. Extensive results on the KITTI, 3DMatch, and 3DLoMatch datasets demonstrate that our method achieves state-of-the-art performance compared to both traditional and learning-based methods in various indoor and outdoor scenes.

## 1 Introduction

Point cloud registration is a fundamental task in vision and robotics, playing an important role in many applications such as 3D perception and reconstruction, simultaneous localization and mapping (SLAM), and autonomous driving [38, 45, 33]. It aims to align two partially overlapping point clouds by estimating a rigid transformation between them. A common registration pipeline involves extracting features through 3D local descriptors, establishing correspondences based on feature matching, and estimating the rigid transformation [38, 41]. However, due to the less effectiveness of 3D local descriptors in feature extraction [39], correspondences established through feature matching are prone to outliers, resulting in inaccurate registration.

Recent works in point could registration with outliers can generally be categorized into three groups: learning-based, geometry-only, and optimization-based methods. Learning-based methods [1, 8, 19]

---

use networks to estimate confidence for correspondences and select those with high confidence for transformation estimation. These networks, however, are typically trained on specific scenarios, leading to limited generalization for outlier removal across various datasets [45]. Geometry-only methods [6, 45], such as $SC^2$-PCR [6] and MAC [45], filter out outliers using geometric relations between correspondences. Such methods [6, 45] rely on effective geometric features and may not produce acceptable inlier ratios in complex scenes or noisy environments [18].

On the other hand, optimization-based methods [4, 5, 18, 40, 46] solve the registration problem by formulating some non-convex objectives [18]. The Branch-and-Bound (BnB) algorithm is widely used to solve non-convex objectives [4, 5, 40] due to its ability to guarantee global optimality. However, the efficiency of BnB is affected by the dimensions of search space and the bounds on objectives [18], which may lead to worst-case exponential time [18, 39]. An alternative approach is to relax the non-convex registration problem into a convex semidefinite program [3, 39]. However, semidefinite relaxation is computationally expensive and may introduce outliers or noise, leading to poor estimation results. Therefore, achieving robust and efficient registration in scenarios with high outlier ratios and noise remains a challenging problem.

To address these challenges, we propose a robust inlier identification algorithm for point cloud registration, which reformulates the conventional registration problem as an alignment error $\ell_0$-minimization problem. More specifically, we define the alignment error and formulate an $\ell_0$-minimization problem for each local set, where these sets are built from the compatibility graph of input correspondences. To resolve the non-convex $\ell_0$-minimization problem effectively, we design a two-stage decoupling strategy. First, the alignment error is decoupled into a rotation fitting error and a translation fitting error by calculating the relative positions between points. This decoupling results in two fitting error $\ell_0$-minimization problems with respect to rotation and translation, respectively. Second, null-spaces are introduced to remove rotation or translation from the constraints of fitting error $\ell_0$-minimization problems, thereby decoupling inlier identification from the estimation of rotation or translation. The final decoupled $\ell_0$-minimization problems are solved for fitting errors through Bayes Theorem. For each local set, correspondences with the smallest errors are identified as inliers to generate a transformation hypothesis. The best hypothesis is selected to perform registration.

To the best of our knowledge, we are the first to propose a $\ell_0$-norm based approach to solve the registration problem. We experimentally demonstrate that the proposed algorithm is robust to high outlier ratios and noise, and is efficient with varying numbers of correspondences. Extensive results on the KITTI, 3DMatch, and 3DLoMatch datasets also demonstrate that our method achieves the highest registration accuracy while being competitive in time efficiency compared to state-of-the-art methods. In summary, our main contributions are as follows:

- A novel robust inlier identification algorithm is proposed by reformulating the conventional registration as an alignment error $\ell_0$-minimization problem, which can effectively identify inliers and perform accurate registration under high outlier ratios and noise.

- A two-stage decoupling strategy is designed for the proposed $\ell_0$-minimization problem. This strategy first decouples rotation and translation, and then decouples inlier identification from rotation or translation estimation.

- A robust Bayesian-based approach is proposed to solve the decoupled $\ell_0$-minimization problem and identify inliers, enhancing the algorithm's performance on noisy data.

## 2 Related Work

**3D local descriptors.** Early handcrafted descriptors like PFH [27] and FPFH [26] mainly represent local features by encoding geometric histograms [38]. More recent works attempt to encode 3D local descriptors in a data-driven way. FCGF [9] extracts features through a fully convolutional neural network. Predator [17] applies an attention mechanism to extract salient points in overlapping regions of point clouds. 3DMatch [44] and 3DSmoothNet [14] build a Siamese deep learning architecture for extracting local information. Although these feature descriptors achieve significant performance improvements, it is difficult to establish correspondences that are completely free of outliers [6]. Therefore, robust registration is very important for accurate registration.

**Learning-based methods.** Inspired by the success of deep learning in 3D perception [34, 32, 37, 29], recent works have adopted learning networks for point cloud registration [1, 8, 19, 20, 44]. Deep

global registration (DGR)[8] utilizes sparse convolution and point-by-point MLPs to classify input correspondences. PointDSC[1] explores a spatial consistency-guided non-local inlier classifier to remove outliers. VBReg [19] introduces a variational non-local network for outlier rejection and learns features with Bayesian-driven long-range contextual dependencies. Despite significant advancements in learning-based registration, these methods are designed for specific scenarios and lack generalization across different datasets, which limits their applicability.

**Geometry-only and optimization-based methods.** Traditional methods have shown great value in practical applications because they are generalized and require no training. They are primarily classified into two categories: geometry-only and optimization-based methods. Geometry-only methods [6, 45, 12] rely on geometric relations and graph-theory frameworks to estimate transformations. SC$^2$-PCR [6] uses a second order spatial compatibility measure to compute the similarity between correspondences. MAC [45] loosens the maximum clique constraint to mine more local consensus information in a graph. However, these methods can not guarantee global optimality and may fail under high outlier ratios or noise. Optimization-based methods [46, 39, 5, 19, 35] aim to estimate optimal solutions (in the maximum likelihood sense) for transformations [39]. FGR [46] employs the Geman-McClure cost function and graduated non-convexity to solve the resulting non-convex optimization. Although it is fast and simple, it performs poorly with high outlier ratios [39, 45]. There are also some methods [4, 5, 40] relying on the branch-and-bound (BnB) algorithm for global registration. However, they suffer from high computational complexity and may require exponential time in the worst case [45]. Therefore, achieving robust and efficient registration in scenarios with high outlier ratios and noise remains challenging.

## 3 Methods

### 3.1 Conventional Point Cloud Registration Problem Statement

Given the source point cloud $\mathcal{P} = \{\mathbf{p}_i \in \mathbb{R}^3 \mid i = 1, \ldots, N\}$ and target point cloud $\mathcal{Q} = \{\mathbf{q}_i \in \mathbb{R}^3 \mid i = 1, \ldots, M\}$, the objective of point cloud registration is to align these two point clouds by estimating an optimal rigid transformation $\mathbf{T} = \{\mathbf{R}, \mathbf{t}\}$, where $\mathbf{R} \in \mathrm{SO}(3)$ denotes the rotation matrix and $\mathbf{t} \in \mathbb{R}^3$ denotes the translation vector. The transformation is then solved by the following registration problem [5, 25]:

$$\min_{\mathbf{R}, \mathbf{t}} \sum_{(\mathbf{p}_i, \mathbf{q}_i) \in \mathcal{C}} \|\mathbf{R}\mathbf{p}_i + \mathbf{t} - \mathbf{q}_i\|_2^2 \,, \tag{1}$$

where $\mathcal{C} = \{\mathbf{c}_i \mid i = 1, \ldots, N_c\}$ is the initial correspondence set. Each correspondence $\mathbf{c}_i = (\mathbf{p}_i, \mathbf{q}_i)$ is formed through feature matching, using descriptors extracted from both point clouds.

### 3.2 Inlier Identification via $\ell_0$-minimization

However, the initial correspondence set $\mathcal{C}$ contains a large proportion of outliers due to incorrect feature matching, leading to inaccurate registration. We aim to identify inliers within $\mathcal{C}$ by solving the proposed alignment error $\ell_0$-minimization problem. The pipeline of our method is shown in Fig. 1.

For the input correspondences, we construct a global compatibility graph using the second-order compatibility measure [6], where correspondences are represented as nodes and edges link geometrically compatible nodes [45]. Based on the compatibility scores, we select $N_1$ correspondences as reliable seeds, denoted as $\mathcal{C}_s = \{\mathbf{c}_i \mid i = 1, \ldots, N_1\}$. For each seed, we identify $N_2$ compatible correspondences to form a local set [1] (refer to Appendix A.1 for details). The alignment error $\ell_0$-minimization problem is formulated for each local set. Specifically, for correspondences in the $k$-th local set $\{\mathbf{c}_{k_i} = (\mathbf{p}_{k_i}, \mathbf{q}_{k_i}) \mid i = 1, \ldots, N_2\}$, the $\ell_0$-minimization problem is defined as follows:

$$\mathbf{O}_k^* = \arg\min_{\mathbf{O}_k} \|\mathbf{O}_k\|_{\ell_0} \,,$$
$$\text{subject to: } \mathbf{O}_k = \mathbf{Q}_k - \mathbf{P}_k \mathbf{R}_k - \mathbf{t}_k \mathbf{1}^T - \mathbf{\Xi}_k \,, \tag{2}$$

where $\mathbf{P}_k = \{\mathbf{p}_k\} \in \mathbb{R}^{N_2 \times 3}$ and $\mathbf{Q}_k = \{\mathbf{q}_k\} \in \mathbb{R}^{N_2 \times 3}$ denote the source and target points in the $k$-th local set. $\mathbf{O}_k$ represents the introduced alignment error, $\mathbf{\Xi}_k$ represents the Gaussian noise, and $\mathbf{1}$ is a column vector of ones, ensuring the translation vector $\mathbf{t}$ is applied to each point in $\mathbf{P}_k$. The alignment error $\mathbf{O}_k$ also serves as an inlier indicator. If the $i$-th correspondence $\mathbf{c}_{k_i} = (\mathbf{p}_{k_i}, \mathbf{q}_{k_i})$ is

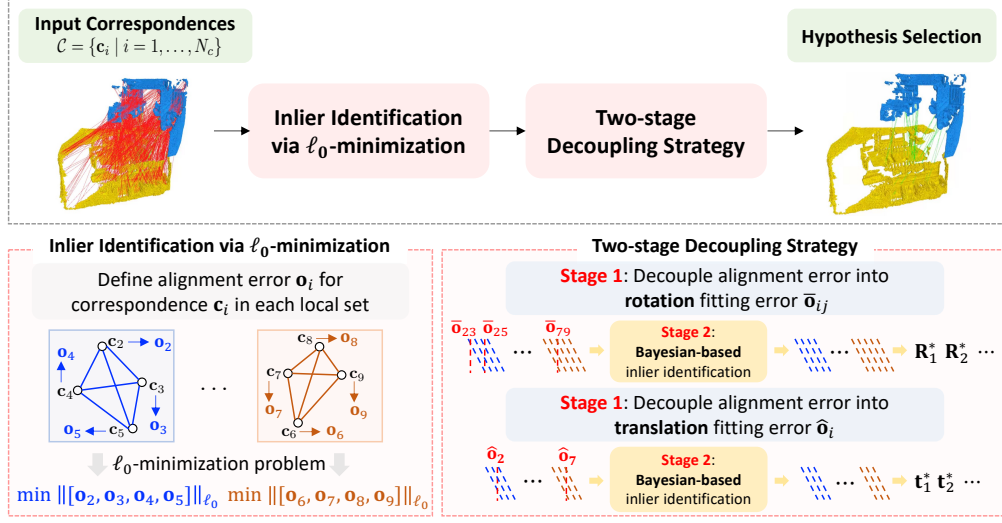

Figure 1: **Pipeline of our method.** 1. Define alignment errors and formulate the $\ell_0$-minimization problem for each local set. 2. Decouple alignment error into rotation and translation fitting errors and decouple inlier identification from the estimation of rotation or translation through the Bayes Theorem. 3. Select the best hypothesis for registration.

an inlier (*i.e.,* it satisfies $|\mathbf{R}\mathbf{p}_{k_i} + \mathbf{t} - \mathbf{q}_{k_i}| \leq \xi_{k_i}$ with the Gaussian noise $\xi_{k_i}$), $\mathbf{o}_{k_i}$ should ideally be a zero vector. Consequently, the indices of zero vectors in the solution $\mathbf{O}_k^*$ of Eq. (2) correspond to the inlier indices in the $k$-th set. The formulations for other local sets are defined in a similar way.

A key insight into our approach is the use of $\ell_0$ norm to optimize alignment errors. This is based on the principle that only inliers can be fitted by the same transformation [5], and the optimal transformation is estimated as the one that fits the largest number of inlier correspondences. Therefore, our optimization objective is to maximize the count of zero vectors in the alignment error. Compared to the common formulations for point cloud registration [18], such as consensus maximization [5, 7] and truncated least-squares [39], our formulation reduces the impact of outliers through $\ell_0$ norm. The focus of this norm is to minimize the number of non-zero vectors rather than their magnitudes, thereby enhancing the robustness of our method to outliers and noise.

### 3.3 Two-stage Decoupling Strategy

To resolve the proposed $\ell_0$-minimization problem, we design a two-stage decoupling strategy. The solution process is described for the $k$-th local set and similarly applied to other local sets.

**Decoupling the alignment error into rotation and translation fitting errors.** Simultaneously estimating the rigid transformation with 6 degrees of freedom (DOF) is time-consuming due to the high-dimensional parameter space [5, 39]. To effectively resolve the $\ell_0$-minimization problem proposed in Eq. (2) for each local set, we decouple the 6-DOF transformation into 3-DOF rotation and 3-DOF translation by computing the relative positions between point pairs. For any two given points $\mathbf{p}_{k_i}$ and $\mathbf{q}_{k_j}$ in the $k$-th local set, the translation vector $\mathbf{t}_k$ cancels out in the subtraction [39]:

$$\mathbf{q}_{k_j} - \mathbf{q}_{k_i} = \mathbf{R}_k \left(\mathbf{p}_{k_j} - \mathbf{p}_{k_i}\right) + \left(\mathbf{o}_{k_j} - \mathbf{o}_{k_i}\right) + \left(\xi_{k_j} - \xi_{k_i}\right) . \tag{3}$$

Based on Eq. (3), we define $\bar{\mathbf{q}}_{k_{ij}} = \mathbf{q}_{k_j} - \mathbf{q}_{k_i}$ and $\bar{\mathbf{p}}_{k_{ij}} = \mathbf{p}_{k_j} - \mathbf{p}_{k_i}$ as the relative positions. $\bar{\mathbf{o}}_{k_{ij}} = \mathbf{o}_{k_j} - \mathbf{o}_{k_i}$ represents the rotation fitting error to minimize, unaffected by translation. $\bar{\xi}_{k_{ij}} = \xi_{k_j} - \xi_{k_i}$ is the Gaussian noise. If both the $i$-th and $j$-th correspondences are inliers, $\bar{\mathbf{o}}_{k_{ij}}$ should ideally be a zero vector. Therefore, the rotation fitting error $\bar{\mathbf{o}}_{k_{ij}}$ for the correspondence pair $\mathbf{c}_{k_i} = (\mathbf{p}_{k_i}, \mathbf{q}_{k_i})$ and $\mathbf{c}_{k_j} = (\mathbf{p}_{k_j}, \mathbf{q}_{k_j})$ is formulated as:

$$\bar{\mathbf{o}}_{k_{ij}} = \bar{\mathbf{q}}_{k_{ij}} - \mathbf{R}_k \bar{\mathbf{p}}_{k_{ij}} - \bar{\xi}_{k_{ij}} . \tag{4}$$

Having decoupled rotation from translation, we can now formulate the $\ell_0$-minimization problem for the rotation fitting error $\bar{\mathbf{O}}_k$ in the $k$-th local set, focusing on the 3-DOF rotation $\mathbf{R}_k$:

$$\bar{\mathbf{O}}_k^* = \arg\min_{\bar{\mathbf{O}}_k} \|\bar{\mathbf{O}}_k\|_{\ell_0} ,$$
$$\text{subject to: } \bar{\mathbf{O}}_k = \bar{\mathbf{Q}}_k - \bar{\mathbf{P}}_k \mathbf{R}_k - \bar{\mathbf{\Xi}}_k , \tag{5}$$

where $\bar{\mathbf{Q}}_k \in \mathbb{R}^{\bar{N} \times 3}$ and $\bar{\mathbf{P}}_k \in \mathbb{R}^{\bar{N} \times 3}$ are relative positions between all point pairs in $\mathbf{Q}_k$ and $\mathbf{P}_k$, respectively. Here, $\bar{N} = \frac{N_2(N_2-1)}{2}$ is the number of relative point pairs in a local set. The Gaussian noise $\bar{\mathbf{\Xi}}_k$ is modeled as $\mathcal{N}(0, \lambda_R \mathbf{I})$, where $\lambda_R$ indicates the variance.

Once obtaining the rotation estimate $\mathbf{R}_k^*$ by solving Eq. (5), we can substitute it back into Eq. (2) to estimate the translation. The $\ell_0$-minimization problem for the translation fitting error $\hat{\mathbf{O}}_k$ is formulated as follows, focusing on the 3-DOF translation $\mathbf{t}_k$:

$$\hat{\mathbf{O}}_k^* = \arg\min_{\hat{\mathbf{O}}_k} \|\hat{\mathbf{O}}_k\|_{\ell_0} ,$$
$$\text{subject to: } \hat{\mathbf{O}}_k = \mathbf{Q}_k - \mathbf{P}_k \mathbf{R}_k^* - \mathbf{t}_k \mathbf{1}^T - \mathbf{\Xi}_k , \tag{6}$$

where $\mathbf{\Xi}_k$ is modeled as $\mathcal{N}(0, \lambda_t \mathbf{I})$, where $\lambda_t$ indicates the variance of Gaussian noise. The translation $\mathbf{t}_k^*$ is estimated by solving Eq. (6).

**Decoupling rotation estimation from $\ell_0$-minimization.** Optimizing the estimation of rotation while simultaneously identifying inliers is a chicken-and-egg problem, because reliable identification of inliers depends on the precise rotation estimation (as shown in Eq. (5)). To address this, we further decouple inlier identification from the estimation of rotation. The inliers that can be fitted by the same rotation are identified through Bayes Theorem and used for the subsequent rotation estimation.

We incorporate a robust Bayesian approach to solve Eq. (5), improving the algorithm's robustness to noisy data [42]. The key step is to define a null-space matrix $\bar{\mathbf{\Theta}}_k$, whose rows form a basis for the left null space of $\bar{\mathbf{P}}_k$. By left-multiplying each term in the constraint of Eq. (5) with $\bar{\mathbf{\Theta}}_k$, the component associated with the rotation $\mathbf{R}_k$ is eliminated:

$$\bar{\mathbf{\Theta}}_k \bar{\mathbf{O}}_k = \bar{\mathbf{\Theta}}_k \bar{\mathbf{Q}}_k - \bar{\mathbf{\Theta}}_k \bar{\mathbf{\Xi}}_k , \tag{7}$$

where $\bar{\mathbf{\Theta}}_k \bar{\mathbf{O}}_k$ represents the transformed rotation fitting error. Given that $\bar{\mathbf{\Xi}}_k$ is Gaussian noise and the left-multiplication by $\bar{\mathbf{\Theta}}_k$ is a linear operation, $\bar{\mathbf{\Theta}}_k \bar{\mathbf{\Xi}}_k$ also follows a Gaussian distribution with a covariance matrix of $\lambda_R \bar{\mathbf{\Theta}}_k \bar{\mathbf{\Theta}}_k^T$. The likelihood is formulated as:

$$P(\tilde{\bar{\mathbf{Q}}}_k \mid \bar{\mathbf{O}}_k) = \mathcal{N}(\bar{\mathbf{\Theta}}_k \bar{\mathbf{O}}_k, \lambda_R \bar{\mathbf{\Pi}}_k) \propto \exp\left[ -\frac{1}{2\lambda_R} \left\| (\tilde{\bar{\mathbf{Q}}}_k - \bar{\mathbf{\Theta}}_k \bar{\mathbf{O}}_k)^T \bar{\mathbf{\Pi}}_k^{-1} (\tilde{\bar{\mathbf{Q}}}_k - \bar{\mathbf{\Theta}}_k \bar{\mathbf{O}}_k) \right\|_F^2 \right] , \tag{8}$$

where $\tilde{\bar{\mathbf{Q}}}_k = \bar{\mathbf{\Theta}}_k \bar{\mathbf{Q}}_k$ and $\bar{\mathbf{\Theta}}_k \bar{\mathbf{\Theta}}_k^T = \bar{\mathbf{\Pi}}_k$. Based on the Bayes Theorem and Maximum A Posteriori (MAP) estimate, the unconstrained optimization for rotation fitting error $\ell_0$-minimization in Eq. (5) is redefined as:

$$\min_{\bar{\mathbf{O}}_k} \frac{1}{2} \left\| (\tilde{\bar{\mathbf{Q}}}_k - \bar{\mathbf{\Theta}} \bar{\mathbf{O}}_k)^T \bar{\mathbf{\Pi}}_k^{-1} (\tilde{\bar{\mathbf{Q}}}_k - \bar{\mathbf{\Theta}}_k \bar{\mathbf{O}}_k) \right\|_F^2 + \lambda_R \left\| \bar{\mathbf{O}}_k \right\|_{\ell_0}^2 , \tag{9}$$

where $\lambda_R$ is the regularization parameter that trades off the fitting error and model complexity. However, since the formulation incorporating the $\ell_0$ norm is known to be computationally expensive, we use the following convex relaxation:

$$\min_{\bar{\mathbf{O}}_k} \frac{1}{2} \left\| (\tilde{\bar{\mathbf{Q}}}_k - \bar{\mathbf{\Theta}} \bar{\mathbf{O}}_k)^T \bar{\mathbf{\Pi}}_k^{-1} (\tilde{\bar{\mathbf{Q}}}_k - \bar{\mathbf{\Theta}}_k \bar{\mathbf{O}}_k) \right\|_F^2 + \lambda_R \left\| \bar{\mathbf{O}}_k \right\|_F^2 , \tag{10}$$

where $\|\cdot\|_F$ is the Frobenius norm ($F$ norm), which is both differentiable and convex. To find the optimal solution, we set the gradient of the objective function with respect to $\bar{\mathbf{O}}_k$ to zero:

$$-\bar{\mathbf{\Theta}}_k^T \bar{\mathbf{\Pi}}_k^{-1} (\tilde{\bar{\mathbf{Q}}}_k - \bar{\mathbf{\Theta}}_k \bar{\mathbf{O}}_k) + 2\lambda_R \bar{\mathbf{O}}_k = 0 . \tag{11}$$

The optimal explicit solution $\bar{\mathbf{O}}_k^*$ can be directly calculated as:

$$\bar{\mathbf{O}}_k^* = (\bar{\mathbf{\Theta}}_k^T \bar{\mathbf{\Pi}}_k^{-1} \bar{\mathbf{\Theta}}_k + 2\lambda_R \mathbf{I})_k^{-1} \bar{\mathbf{\Theta}}_k^T \bar{\mathbf{\Pi}}_k^{-1} \tilde{\bar{\mathbf{Q}}}_k . \tag{12}$$

Based on $\bar{\mathbf{O}}_k^*$, we identify top-$K_R$ correspondences with minimal rotation fitting error for accurate rotation estimation. These correspondences, indexed by $\mathcal{I}_R$, provide the basis for estimating rotation from the SVD decomposition of the matrix $H = U\Sigma V^T \in \mathbb{R}^{3\times3}$ [23]. For the $k$-th local set, the rotation hypothesis is estimated as [5, 2]:

$$H = \sum_{(i,j)\in\mathcal{I}_R} \bar{\mathbf{P}}_{k_i}\bar{\mathbf{Q}}_{k_j}^T, \quad \mathbf{R}_k^* = U\,\mathrm{diag}\left(1, 1, \det\left(UV^T\right)\right)V. \tag{13}$$

**Decoupling translation estimation from $\ell_0$-minimization.** Employing a strategy similar to that used for rotation estimation, we utilize a null-space matrix $\mathbf{\Theta}_k$ that satisfies $\mathbf{\Theta}_k\mathbf{1} = \mathbf{0}$ to isolate the translation. By applying $\mathbf{\Theta}_k$ to the transpose of the constraint in Eq. (6), we eliminate the components associated with translation $\mathbf{t}_k$:

$$\mathbf{\Theta}_k\hat{\mathbf{O}}_k^T = \mathbf{\Theta}_k(\mathbf{Q}_k - \mathbf{P}_k\mathbf{R}_k^*)^T - \mathbf{\Theta}_k\mathbf{\Xi}_k^T. \tag{14}$$

Incorporating the Bayes Theorem, we formulate the following convex relaxation for the unconstrained optimization problem:

$$\min_{\hat{\mathbf{O}}_k} \frac{1}{2}\left\|(\mathbf{X}_k - \mathbf{\Theta}_k\hat{\mathbf{O}}_k^T)^T\mathbf{\Pi}_k^{-1}(\mathbf{X}_k - \mathbf{\Theta}_k\hat{\mathbf{O}}_k^T)\right\|_F^2 + \lambda_t\left\|\hat{\mathbf{O}}_k\right\|_F^2, \tag{15}$$

where $\mathbf{X}_k = \mathbf{\Theta}_k(\mathbf{Q}_k^T - (\mathbf{P}_k\mathbf{R}_k^*)^T)$ and $\mathbf{\Pi}_k = \mathbf{\Theta}_k\mathbf{\Theta}_k^T$. The explicit solution $\hat{\mathbf{O}}_k^*$ is obtained by solving the gradient of the objective function with respect to $\hat{\mathbf{O}}_k$:

$$\hat{\mathbf{O}}_k^* = ((2\lambda_t\mathbf{I} + \mathbf{\Theta}_k^T\mathbf{\Pi}_k^{-1}\mathbf{\Theta}_k)^{-1}\mathbf{\Theta}_k^T\mathbf{\Pi}_k^{-1}\mathbf{X}_k)^T, \tag{16}$$

where $\mathbf{I}$ denotes the identity matrix. Using $\hat{\mathbf{O}}_k^*$, top-$K_t$ correspondences with the smallest errors are identified as inliers for translation estimation. Their index set is denoted as $\mathcal{I}_t$. The translation hypothesis $\mathbf{t}_k^*$ for the $k$-th local set is estimated based on these inliers $(\mathbf{p}_{k_i}, \mathbf{q}_{k_j})$, with $(i, j) \in \mathcal{I}_t$.

### 3.4 Hypothesis Selection

Finally, we evaluate and select the best estimation from the transformation hypotheses computed for all local sets:

$$(\mathbf{R}^*, \mathbf{t}^*) = \arg\max_{\mathbf{R}_k^*, \mathbf{t}_k^*} \sum_{i=1}^N [\|\mathbf{R}_k^*\mathbf{p}_i + \mathbf{t}_k^* - \mathbf{q}_i\|_2 < \tau], \tag{17}$$

where $N_c$ is the number of input initial correspondences and $\tau$ is a predefined error threshold. For each transformation hypothesis, we quantify its effectiveness by counting the number of correspondences that satisfy the constraints within $\tau$. The transformation with the highest inlier count is selected for registration.

## 4 Experiments

### 4.1 Datasets and Experimental Setup

**Synthetic dataset.** We evaluate the accuracy, robustness, and efficiency of our algorithm using the Bunny point cloud from the Stanford 3D Scan Repository [10]. Similar to [5, 39], the Bunny model is downsampled to $N_c$ points and resized to fit a $[0, 1]^3$ cube, creating the source point cloud $\mathcal{P}$. To generate the target point cloud $\mathcal{Q}$, a random transformation $(\mathbf{R}, \mathbf{t})$ is applied to $\mathcal{P}$ and then Gaussian noise $\epsilon_i \sim \mathcal{N}(0, \sigma^2\mathbf{I}_3)$ is added. A pair of the original and moved points defines an inlier. The inliers are contaminated with outliers generated by random transformations.

**Outdoor scenes.** For evaluations on outdoor scenes, we conduct experiments on the KITTI dataset [13]. Following [5, 6], we use 555 pairs of point clouds from scenes 8 to 10 for testing. We construct a 30cm voxel grid to downsample point clouds and form correspondences using handcrafted FPFH [26] and learned FCGF [9] descriptors.

**Indoor scenes.** We conduct experiments on the 3DMatch dataset [44] to evaluate performance on indoor scenes. Following [5, 6, 45], we use RGB-D scans from 8 real indoor scenes for testing. The

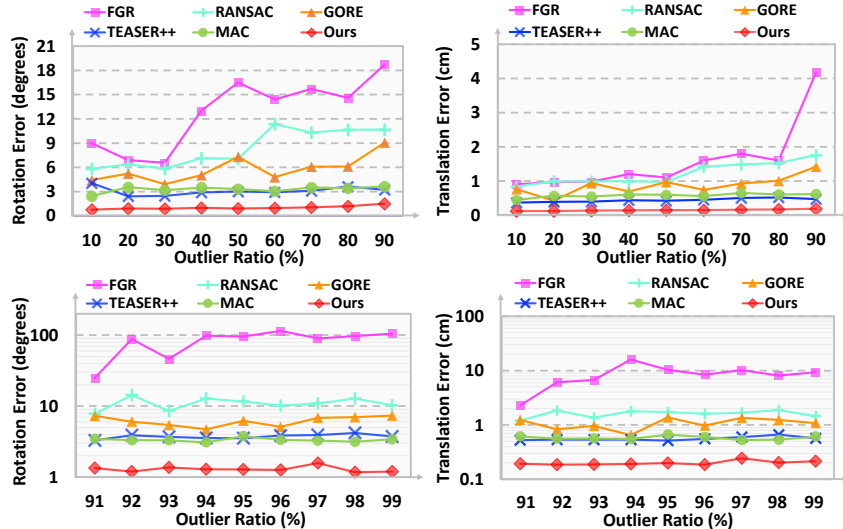

Figure 2: **Robustness to outliers.** The first row compares the rotation and translation errors as the outlier ratio increases from $10\%$ to $90\%$ on the Bunny dataset [10], while the second row focuses on the scenarios of extreme outliers, *i.e.,* the outlier ratio varies from $91\%$ to $99\%$. Our method demonstrates to be more robust to outliers compared to other methods [4, 12, 39, 45, 46].

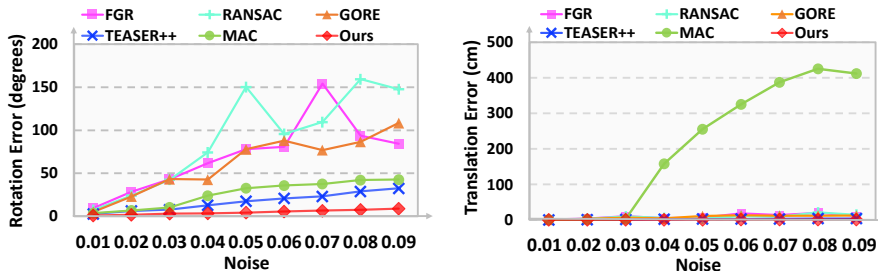

Figure 3: **Robustness to noise.** Comparison results with [4, 12, 39, 45, 46] as the noise standard deviation increases from $0.01$ to $0.09$ on the Bunny dataset [10].

point clouds are downsampled using a 5cm voxel grid. We use the hand-crafted FPFH [26] along with two learned descriptors, FCGF [26] and 3DSmoothNet [14], for feature extraction. To evaluate our method in more challenging scenarios, we conduct experiments on 3DLoMatch [17] (overlap rate between scenes $< 30\%$). Following [5, 19], the Predator descriptor [17] is used in 3DLoMatch.

**Evaluation criteria.** Following [5, 39], we use the rotation error (RE), translation error (TE), and registration recall (RR) as evaluation metrics. The registration is considered successful when the $RE \le 15°$, $TE \le 30cm$ on 3DMatch & 3DLoMatch datasets, and $RE \le 5°$, $TE \le 60cm$ on KITTI dataset. Average RE and TE are computed only on the successfully registered pairs [5, 6].

**Implementation details.** We implement our method in PyTorch [24]. All the experiments are conducted on a machine with an Intel Xeon Gold 6134 CPU and a single NVIDIA GTX3090.

## 4.2 Evaluation on Synthetic Dataset

**Robustness to outliers.** We evaluate the robustness to outliers by increasing the outlier ratio from $10\%$ to $90\%$. The Bunny point cloud is downsampled to $N_c = 500$. We add zero-mean Gaussian noise with a standard deviation set to $\sigma = 0.01$. For each outlier ratio, we conduct $50$ independent trials and report the average rotation error (RE) and translation error (TE). We compare our method with state-of-the-art traditional methods [4, 12, 39, 45, 46]. As shown in the first row of Fig. 2, the rotation and translation errors of FGR [46] increase sharply as the proportion of outliers increases. RANSAC [12] and GORE [4] start failing at an outlier ratio of $60\%$. Our method remains robust to outliers up to $90\%$ and produces more accurate estimates than all other methods. We further

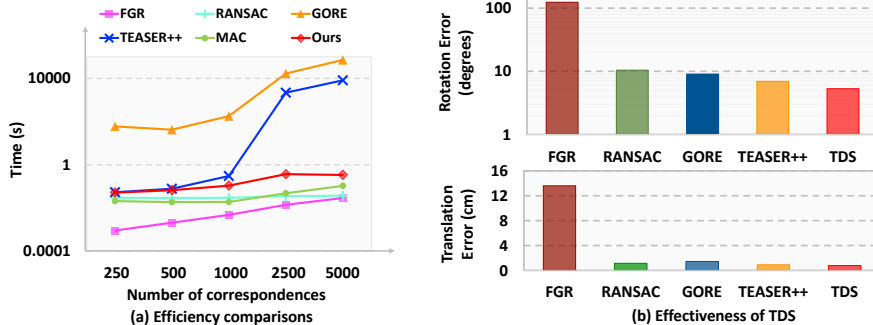

Figure 4: **Efficiency and effectiveness.** The experiment results on the Bunny dataset [10]. (a) Efficiency comparison with other methods [4, 12, 39, 45, 46] with respect to the number of correspondences. (b) Comparison of the proposed two-stage decoupling strategy (TDS) with optimization-based methods at an outlier ratio of 90%.

Table 1: Comparison results on KITTI dataset [13] using the FPFH [26] and FCGF [9] descriptors.

| | FPFH | | | FCGF | | | Time(s) |
|---|---|---|---|---|---|---|---|
| | RR(%)↑ | RE(°)↓ | TE(cm)↓ | RR(%)↑ | RE(°)↓ | TE(cm)↓ | |
| *i) Traditional* | | | | | | | |
| FGR [46] | 5.23 | 0.86 | 43.84 | 89.54 | 0.46 | 25.72 | 3.88 |
| RANSAC [12] | 74.41 | 1.55 | 30.20 | 80.36 | 0.73 | 26.79 | 5.43 |
| TEASER++ [39] | 91.17 | 1.03 | 17.98 | 95.51 | 0.33 | 22.38 | 0.03 |
| SC$^2$-PCR [6] | 99.46 | 0.35 | 7.87 | 98.02 | 0.33 | 20.69 | 0.31 |
| MAC [45] | 97.66 | 0.41 | 8.61 | 97.84 | 0.34 | 19.34 | 3.29 |
| TR-DE [5] | 96.76 | 0.90 | 15.63 | **98.20** | 0.38 | **18.00** | - |
| TEAR [18] | 99.10 | 0.39 | 8.62 | - | - | - | - |
| *ii) Deep learned* | | | | | | | |
| DGR [8] | 77.12 | 1.64 | 33.10 | 96.90 | 0.34 | 21.70 | 2.29 |
| PointDSC [1] | 98.92 | 0.38 | 8.35 | 97.84 | 0.33 | 20.32 | 0.45 |
| VBReg [19] | 98.92 | 0.45 | 8.41 | 98.02 | 0.32 | 20.91 | 0.24 |
| Ours | **99.56** | **0.34** | **7.85** | **98.20** | **0.32** | 20.73 | 0.54 |

compare the performance of different methods under extreme outlier ratios, *i.e.,* when the outlier ratio increases from 91% to 99%. The second row of Fig. 2 shows that even with outlier ratios as high as 99%, our method continues to perform well, consistently producing lower transformation errors than other methods.

**Robustness to noise.** We further evaluate the robustness against Gaussian noise with different variances. As the noise standard deviation increases from $\sigma = 0.01$ to $\sigma = 0.1$, the geometric structure of the Bunny model is completely destroyed [39] (refer to Appendix A.5). Fig. 3 shows the comparison results as $\sigma$ increases from 0.01 to 0.09. When the noise variance reaches 0.03, the translation errors of geometric-only method MAC [45] significantly increase. Both FGR [46] and RANSAC [12] show large rotation errors when $\sigma$ increases to 0.05. In contrast, our method achieves the lowest rotation and translation errors under high noise, demonstrating its robustness to noise.

**Efficiency and accuracy.** We increase the number of correspondences $N_c$ from 250 to 5000 to compare efficiency and accuracy. We set the noise standard deviation $\sigma$ to 0.01 and the outlier ratio to 50%. The comparison results are shown in Fig. 4(a). As the number of correspondences increases, the running time of GORE [4] and TEASER++ [39] increases significantly. Notably, when $N_c$ grows to 2500, the running time of GORE is about $10^4$ times longer than that of our method. Our method solves the $\ell_0$-minimization problems with explicit solutions, significantly enhancing efficiency through parallel matrix computations and GPU execution. The curves of FGR, RANSAC, MAC, and our method in Fig. 4(a) are flat and difficult to visually distinguish, indicating the efficiency of these methods. However, as shown in Appendix A.6, our method outperforms FGR [46], RANSAC [12], and MAC [45] in registration accuracy. Therefore, our inlier identification algorithm via $\ell_0$-minimization is efficient while maintaining high accuracy.

**Effectiveness of the two-stage decoupling strategy.** We evaluate the effectiveness of the two-stage decoupling strategy (TDS) by formulating the $\ell_0$-minimization problem directly on the Bunny data instead of local sets and estimating the final rotation and translation. Specifically, we set $N_c = 100$ and $\sigma = 0.01$. As shown in Fig. 4(b), we compare the TDS with optimization-based

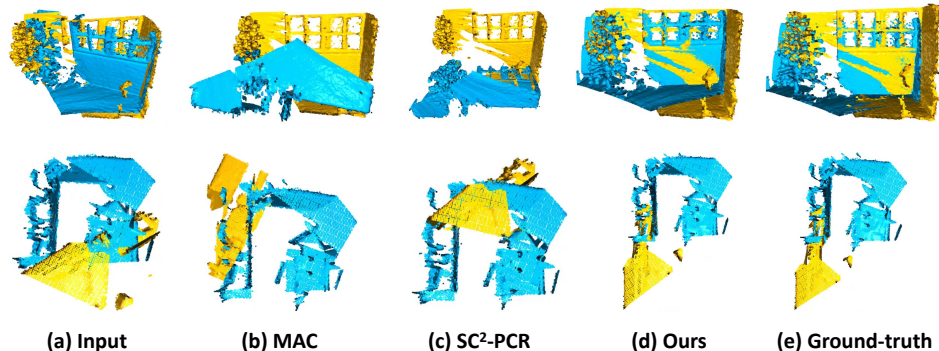

**(a) Input**     **(b) MAC**     **(c) SC²-PCR**     **(d) Ours**     **(e) Ground-truth**

Figure 5: **Qualitative comparisons with other methods.** Qualitative comparisons on the 3DMatch (the first row) and 3DLoMatch (the second row) datasets.

Table 2: Comparisons results on 3DMatch [44] using FPFH, FCGF, and 3DSmoothNet descriptors.

| | FPFH | | | FCGF | | | 3DSmoothNet | | | Time(s) |
|---|---|---|---|---|---|---|---|---|---|---|
| | RR(%)↑ | RE(°)↓ | TE(cm)↓ | RR(%)↑ | RE(°)↓ | TE(cm)↓ | RR(%)↑ | RE(°)↓ | TE(cm)↓ | |
| *i) Traditional* | | | | | | | | | | |
| FGR [46] | 40.91 | 4.96 | 10.25 | 78.93 | 2.90 | 8.41 | 73.26 | 2.51 | 7.45 | 0.89 |
| RANSAC [12] | 66.10 | 3.95 | 11.03 | 91.44 | 2.69 | 8.38 | 92.30 | 2.59 | 7.91 | 2.86 |
| TEASER++ [39] | 75.48 | 2.48 | 7.31 | 85.71 | 2.73 | 8.66 | 92.05 | 2.23 | 6.62 | 0.03 |
| SC²-PCR [6] | 83.90 | 2.12 | 6.69 | 93.16 | 2.06 | 6.53 | 94.82 | 1.76 | 5.98 | 0.12 |
| MAC [45] | 83.90 | **2.11** | 6.80 | **93.72** | **2.04** | 6.54 | 94.57 | 2.21 | 6.52 | 5.54 |
| TR-DE [5] | - | - | - | - | - | - | 91.37 | 2.71 | 7.62 | - |
| TEAR [18] | - | - | - | - | - | - | 94.52 | 2.06 | 6.55 | - |
| *ii) Deep learned* | | | | | | | | | | |
| DGR [8] | 32.84 | 2.45 | 7.53 | 88.85 | 2.28 | 7.02 | - | - | - | 1.53 |
| PointDSC [1] | 72.95 | 2.18 | **6.45** | 91.87 | 2.10 | 6.54 | 93.65 | 2.17 | 6.75 | 0.10 |
| VBReg [19] | 82.57 | 2.14 | 6.77 | 93.53 | **2.04** | 6.49 | 37.09 | 6.15 | 15.65 | 0.20 |
| Ours | **83.92** | 2.12 | 6.64 | 93.28 | **2.04** | **6.48** | **95.07** | **1.75** | **5.97** | 0.36 |

methods [4, 12, 39, 46] at an outlier ratio of 90%. Our TDS achieves the highest registration accuracy, demonstrating its inlier identification capability. Additional competitive results as the outlier ratio increases from 0% to 90% are provided in Appendix A.7.

## 4.3 Evaluation on Outdoor Scenes

To evaluate our algorithm on real outdoor scenes, we conduct experiments on the KITTI dataset [13]. The comparison results with state-of-the-art traditional [5, 6, 12, 18, 39, 45, 46] and learning-based [1, 8, 19] methods are reported in Table 1. We first use the FPFH [26] descriptor to generate initial correspondences. As shown in the left column of Table 2, our method outperforms traditional and learning-based methods on all metrics. For the most important criterion of registration recall (RR), our method improves by about 2% over the nearest competitor MAC [45]. Following [6], the average RE and TE are only calculated on successfully registered pairs, which makes methods with high registration recall more likely to have larger average errors. Nonetheless, our method still achieves the best results on RE and TE. Besides, we report the comparison results with the FCGF [9] descriptor in the right column of Table 2. Our method achieves the highest RR and the lowest RE due to its effective inlier identification algorithm. The superior performance demonstrates the ability of our method to align sparse and non-uniformly distributed data in outdoor scenes. In addition to its high accuracy, our method also achieves comparable efficiency, making it highly competitive for practical applications. The visualizations of registration results on KITTI are provided in Appendix A.12.

## 4.4 Evaluation on Indoor Scenes

We further conduct experiments on the 3DMatch [44] and 3DLoMatch [17] datasets to evaluate the performance in real indoor scenes. The comparison results are reported in Table 2 and Table 3.

**Combined with FPFH, FCGF, and 3DSmoothNet descriptors.** As shown in the left column of Table 2, compared to both traditional and learning-based methods, our method achieves the highest RR with the handcrafted FPFH [26] descriptor. The middle column of Table 2 reports the comparison results with the learned FCGF [9] descriptor. Our method achieves the lowest RE and TE. Compared to SC²-PCR [6], our method improves the RR, RE, and TE by 0.13%, 0.97%, and 0.77% respectively,

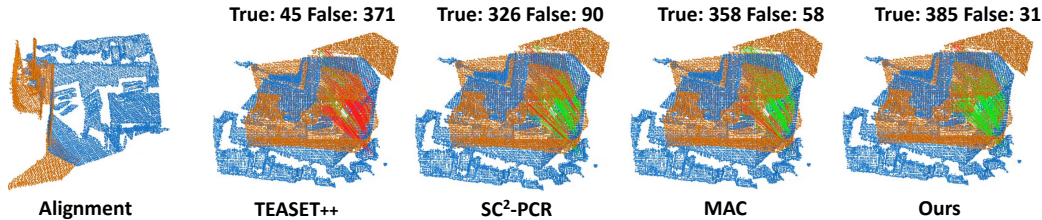

| True: 45 False: 371 | True: 326 False: 90 | True: 358 False: 58 | True: 385 False: 31 |
| Alignment | TEASET++ | SC²-PCR | MAC | Ours |

Figure 6: **Comparison results on output inlier ratio.** We compare the predicted inlier counts of correct and incorrect correspondences in 3DLoMatch [17]. The first column provides the ground truth alignment, which shows that overlap is very limited. The significantly larger inlier ratio can be observed from the incorrect (red lines) and correct (green lines) correspondences.

which benefit from our $\ell_0$-minimization formulation for inlier identification. Since TR-DE [5] and TEAR [18] have not made their code or results for FPFH and FCGF publicly available, their results are excluded in the left and middle columns of Table 2. Following [18], we also compare the registration accuracy using the learned 3DSmoothNet [14] to extract features. The results in the right column of Table 2 show that our method outperforms all other methods across all evaluation metrics, demonstrating the robustness of our method to different local descriptors. We show the results of qualitative comparisons in Fig. 5 and Appendix A.12. Methods such as MAC may fail in scenes with ambiguous features or limited overlap. Our algorithm still achieves satisfactory alignment and is close to the ground truth.

**Robust to lower overlap ratios.** Furthermore, we report results for a more challenging dataset: 3DLoMatch [17] (overlap rate < 30%). Following [5, 19], we use the Predator [17] descriptor to generate the initial correspondences. We compare the registration recall (RR) under different numbers of correspondences. As shown in Table 3, the proposed method improves the average RR by 7% over TR-DE [5], demonstrating the effectiveness of our method in dealing with

Table 3: Registration rate on the 3DLoMatch dataset [17] with different number of correspondences.

|  | 5000 | 2500 | 1000 | 500 | 250 |
| --- | --- | --- | --- | --- | --- |
| Predator | | | | | |
| FGR [46] | 36.4 | 38.2 | 39.7 | 39.6 | 38.0 |
| RANSAC [12] | 62.3 | 62.8 | 62.4 | 61.5 | 58.2 |
| TEASER++ [39] | 62.9 | 62.6 | 61.9 | 59.0 | 56.7 |
| SC²-PCR [6] | 68.9 | 68.4 | _68.7_ | _67.1_ | _64.9_ |
| MAC [45] | _69.4_ | 69.3 | 68.4 | **67.7** | 64.6 |
| TR-DE [5] | 64.0 | 64.8 | 61.7 | 58.8 | 56.5 |
| PointDSC [1] | 68.1 | 67.3 | 66.5 | 63.4 | 60.5 |
| VBReg [19] | **69.9** | _69.8_ | _68.7_ | 66.4 | 63.0 |
| Ours | **69.9** | **69.9** | **69.2** | **67.7** | **65.0** |

low-overlap scenarios. In Fig. 6, we compare the output inlier ratio with traditional methods [6, 39, 45] in the low overlap scenario. Our method is more effective with more correct predicted inliers.

## 5 Conclusion

In this paper, we propose a robust inlier identification algorithm by reformulating the conventional registration problem as an alignment error $\ell_0$-minimization problem. For each local set, we resolve the $\ell_0$-minimization problem using a designed two-stage decoupling strategy. First, the alignment error is decoupled to a rotation fitting error and a translation fitting error, formulating two decoupled $\ell_0$-minimization problems. Second, null-space matrices are introduced to decouple the inlier identification from the estimation of rotation or translation respectively, there by applying a robust Bayesian approach to decoupled $\ell_0$-minimization problems and solving for fitting errors. Correspondences with the smallest errors are identified as inliers to generate a transformation hypothesis for each local set. We experimentally demonstrate that the proposed algorithm is robust to high outlier ratios and noise. Extensive results on the KITTI, 3DMatch, and 3DLoMatch datasets also demonstrate that our method achieves state-of-the-art registration accuracy while being comparable in efficiency in both indoor and outdoor scenes. Limitations and broader impact are discussed in Appendix A.10.

## 6 Acknowledgements

This work was supported by the National Natural Science Foundation of China (Grant numbers 92167201, 52188102, 62373160).

# References

[1] Bai, X., Luo, Z., Zhou, L., Chen, H., Li, L., Hu, Z., Fu, H., Tai, C.L., 2021. Pointdsc: Robust point cloud registration using deep spatial consistency, in: Proceedings of the IEEE/CVF Conference on Computer Vision and Pattern Recognition, pp. 15859–15869.

[2] Besl, P.J., McKay, N.D., 1992. Method for registration of 3-d shapes, in: Sensor fusion IV: control paradigms and data structures, Spie. pp. 586–606.

[3] Briales, J., Gonzalez-Jimenez, J., 2017. Convex global 3d registration with lagrangian duality, in: Proceedings of the IEEE conference on computer vision and pattern recognition, pp. 4960–4969.

[4] Bustos, A.P., Chin, T.J., 2017. Guaranteed outlier removal for point cloud registration with correspondences. IEEE transactions on pattern analysis and machine intelligence 40, 2868–2882.

[5] Chen, W., Li, H., Nie, Q., Liu, Y.H., 2022a. Deterministic point cloud registration via novel transformation decomposition, in: Proceedings of the IEEE/CVF Conference on Computer Vision and Pattern Recognition, pp. 6348–6356.

[6] Chen, Z., Sun, K., Yang, F., Tao, W., 2022b. Sc2-pcr: A second order spatial compatibility for efficient and robust point cloud registration, in: Proceedings of the IEEE/CVF Conference on Computer Vision and Pattern Recognition, pp. 13221–13231.

[7] Chin, T.J., Suter, D., 2022. The maximum consensus problem: recent algorithmic advances. Springer Nature.

[8] Choy, C., Dong, W., Koltun, V., 2020. Deep global registration, in: Proceedings of the IEEE/CVF conference on computer vision and pattern recognition, pp. 2514–2523.

[9] Choy, C., Park, J., Koltun, V., 2019. Fully convolutional geometric features, in: Proceedings of the IEEE/CVF international conference on computer vision, pp. 8958–8966.

[10] Curless, B., Levoy, M., 1996. A volumetric method for building complex models from range images, in: Proceedings of the 23rd annual conference on Computer graphics and interactive techniques, pp. 303–312.

[11] Dai, A., Nießner, M., Zollhöfer, M., Izadi, S., Theobalt, C., 2017. Bundlefusion: Real-time globally consistent 3d reconstruction using on-the-fly surface reintegration. ACM Transactions on Graphics (ToG) 36, 1.

[12] Fischler, M.A., Bolles, R.C., 1981. Random sample consensus: a paradigm for model fitting with applications to image analysis and automated cartography. Communications of the ACM 24, 381–395.

[13] Geiger, A., Lenz, P., Urtasun, R., 2012. Are we ready for autonomous driving? the kitti vision benchmark suite, in: 2012 IEEE conference on computer vision and pattern recognition, IEEE. pp. 3354–3361.

[14] Gojcic, Z., Zhou, C., Wegner, J.D., Wieser, A., 2019. The perfect match: 3d point cloud matching with smoothed densities, in: Proceedings of the IEEE/CVF conference on computer vision and pattern recognition, pp. 5545–5554.

[15] Guo, J., Wang, Q., Park, J.H., 2020. Geometric quality inspection of prefabricated mep modules with 3d laser scanning. Automation in Construction 111, 103053.

[16] Halber, M., Funkhouser, T., 2017. Fine-to-coarse global registration of rgb-d scans, in: Proceedings of the IEEE Conference on Computer Vision and Pattern Recognition, pp. 1755–1764.

[17] Huang, S., Gojcic, Z., Usvyatsov, M., Wieser, A., Schindler, K., 2021. Predator: Registration of 3d point clouds with low overlap, in: Proceedings of the IEEE/CVF Conference on computer vision and pattern recognition, pp. 4267–4276.

[18] Huang, T., Peng, L., Vidal, R., Liu, Y.H., 2024. Scalable 3d registration via truncated entry-wise absolute residuals. arXiv preprint arXiv:2404.00915 .

[19] Jiang, H., Dang, Z., Wei, Z., Xie, J., Yang, J., Salzmann, M., 2023. Robust outlier rejection for 3d registration with variational bayes, in: Proceedings of the IEEE/CVF Conference on Computer Vision and Pattern Recognition, pp. 1148–1157.

[20] Jiang, Y., Zhou, B., Liu, X., Li, Q., Cheng, C., 2024. Gtinet: Global topology-aware interactions for unsupervised point cloud registration. IEEE Transactions on Circuits and Systems for Video Technology .

[21] Lai, K., Bo, L., Fox, D., 2014. Unsupervised feature learning for 3d scene labeling, in: 2014 IEEE International Conference on Robotics and Automation (ICRA), IEEE. pp. 3050–3057.

[22] Mises, R., Pollaczek-Geiringer, H., 1929. Praktische verfahren der gleichungsauflösung. ZAMM-Journal of Applied Mathematics and Mechanics/Zeitschrift für Angewandte Mathematik und Mechanik 9, 58–77.

[23] Papadopoulo, T., Lourakis, M.I., 2000. Estimating the jacobian of the singular value decomposition: Theory and applications, in: Computer Vision-ECCV 2000: 6th European Conference on Computer Vision Dublin, Ireland, June 26–July 1, 2000 Proceedings, Part I 6, Springer. pp. 554–570.

[24] Paszke, A., Gross, S., Massa, F., Lerer, A., Bradbury, J., Chanan, G., Killeen, T., Lin, Z., Gimelshein, N., Antiga, L., et al., 2019. Pytorch: An imperative style, high-performance deep learning library. Advances in neural information processing systems 32.

[25] Qin, Z., Yu, H., Wang, C., Guo, Y., Peng, Y., Xu, K., 2022. Geometric transformer for fast and robust point cloud registration, in: Proceedings of the IEEE/CVF Conference on Computer Vision and Pattern Recognition, pp. 11143–11152.

[26] Rusu, R.B., Blodow, N., Beetz, M., 2009. Fast point feature histograms (fpfh) for 3d registration, in: 2009 IEEE international conference on robotics and automation, IEEE. pp. 3212–3217.

[27] Rusu, R.B., Blodow, N., Marton, Z.C., Beetz, M., 2008. Aligning point cloud views using persistent feature histograms, in: 2008 IEEE/RSJ international conference on intelligent robots and systems, IEEE. pp. 3384–3391.

[28] Shotton, J., Glocker, B., Zach, C., Izadi, S., Criminisi, A., Fitzgibbon, A., 2013. Scene coordinate regression forests for camera relocalization in rgb-d images, in: Proceedings of the IEEE conference on computer vision and pattern recognition, pp. 2930–2937.

[29] Sun, J., Xie, Y., Chen, L., Zhou, X., Bao, H., 2021. Neuralrecon: Real-time coherent 3d reconstruction from monocular video, in: Proceedings of the IEEE/CVF conference on computer vision and pattern recognition, pp. 15598–15607.

[30] Turk, G., Levoy, M., 1994. Zippered polygon meshes from range images, in: Proceedings of the 21st annual conference on Computer graphics and interactive techniques, pp. 311–318.

[31] Valentin, J., Dai, A., Nießner, M., Kohli, P., Torr, P., Izadi, S., Keskin, C., 2016. Learning to navigate the energy landscape, in: 2016 Fourth International Conference on 3D Vision (3DV), IEEE. pp. 323–332.

[32] Wang, Y., Pan, Z., Li, X., Cao, Z., Xian, K., Zhang, J., 2022. Less is more: Consistent video depth estimation with masked frames modeling, in: Proceedings of the 30th ACM International Conference on Multimedia, pp. 6347–6358.

[33] Wang, Y., Shi, M., Li, J., Hong, C., Huang, Z., Peng, J., Cao, Z., Zhang, J., Xian, K., Lin, G., 2024. Nvds$^{+}$: Towards efficient and versatile neural stabilizer for video depth estimation. IEEE Transactions on Pattern Analysis and Machine Intelligence doi:10.1109/TPAMI.2024.3476387.

[34] Wang, Y., Shi, M., Li, J., Huang, Z., Cao, Z., Zhang, J., Xian, K., Lin, G., 2023. Neural video depth stabilizer, in: Proceedings of the IEEE/CVF International Conference on Computer Vision, pp. 9466–9476.

[35] Wu, Y., Ding, H., Gong, M., Qin, A.K., Ma, W., Miao, Q., Tan, K.C., 2022. Evolutionary multiform optimization with two-stage bidirectional knowledge transfer strategy for point cloud registration. IEEE Transactions on Evolutionary Computation 28, 62–76.

[36] Xiao, J., Owens, A., Torralba, A., 2013. Sun3d: A database of big spaces reconstructed using sfm and object labels, in: Proceedings of the IEEE international conference on computer vision, pp. 1625–1632.

[37] Yang, B., Wen, H., Wang, S., Clark, R., Markham, A., Trigoni, N., 2017. 3d object reconstruction from a single depth view with adversarial learning, in: Proceedings of the IEEE international conference on computer vision workshops, pp. 679–688.

[38] Yang, F., Guo, L., Chen, Z., Tao, W., 2022. One-inlier is first: Towards efficient position encoding for point cloud registration. Advances in Neural Information Processing Systems 35, 6982–6995.

[39] Yang, H., Shi, J., Carlone, L., 2020. Teaser: Fast and certifiable point cloud registration. IEEE Transactions on Robotics 37, 314–333.

[40] Yang, J., Li, H., Jia, Y., 2013. Go-icp: Solving 3d registration efficiently and globally optimally, in: Proceedings of the IEEE International Conference on Computer Vision, pp. 1457–1464.

[41] Yew, Z.J., Lee, G.H., 2022. Regtr: End-to-end point cloud correspondences with transformers, in: Proceedings of the IEEE/CVF Conference on Computer Vision and Pattern Recognition, pp. 6677–6686.

[42] Yuan, Y., Tang, X., Zhou, W., Pan, W., Li, X., Zhang, H.T., Ding, H., Goncalves, J., 2019. Data driven discovery of cyber physical systems. Nature communications 10, 4894.

[43] Yuan, Y., Wu, Y., Fan, X., Gong, M., Ma, W., Miao, Q., 2023. Egst: Enhanced geometric structure transformer for point cloud registration. IEEE Transactions on Visualization and Computer Graphics .

[44] Zeng, A., Song, S., Nießner, M., Fisher, M., Xiao, J., Funkhouser, T., 2017. 3dmatch: Learning the matching of local 3d geometry in range scans, in: CVPR, p. 4.

[45] Zhang, X., Yang, J., Zhang, S., Zhang, Y., 2023. 3d registration with maximal cliques, in: Proceedings of the IEEE/CVF Conference on Computer Vision and Pattern Recognition, pp. 17745–17754.

[46] Zhou, Q.Y., Park, J., Koltun, V., 2016. Fast global registration, in: Computer Vision–ECCV 2016: 14th European Conference, Amsterdam, The Netherlands, October 11-14, 2016, Proceedings, Part II 14, Springer. pp. 766–782.

# A Appendix

In the appendix, we first provide the detailed construction for local sets (Sec. A.1), the rigorous definitions of evaluation metrics (Sec. A.2), then describe the pseudocode for key parts (Sec. A.3) and the hyper-parameter selection (Sec. A.4). We further provide additional experimental results (Sec. A.5, Sec. A.6, Sec. A.7, and Sec. A.8), ablation studies on parameters (Sec. A.9), and discuss the limitations (Sec. A.10) and scalability (Sec. A.11) of our work. Finally, we show more qualitative results of registration on 3DMatch, 3DLoMatch, and KITTI (Sec. A.12) and provide the detailed information for these datasets (Sec. A.13).

## A.1 Local set construction

In this section, we provide the detailed construction for local sets. We first construct a global compatibility graph for input correspondences. Specifically, we calculate the Euclidean distance between the correspondence pair $(\mathbf{c}_i, \mathbf{c}_j)$ as follows:

$$d_{(\mathbf{c}_i,\mathbf{c}_j)} = \big| \|\mathbf{p}_i - \mathbf{p}_j\| - \|\mathbf{q}_i - \mathbf{q}_j\| \big| \,, \tag{18}$$

where $\mathbf{p}_i$ and $\mathbf{p}_j$ denote points in the source point cloud and $\mathbf{q}_i$ and $\mathbf{q}_j$ are the corresponding points in the target point cloud. The first order compatibility score for each pair $(\mathbf{c}_i, \mathbf{c}_j)$ is calculated based on the Euclidean distance, as follows:

$$S_{(\mathbf{c}_i,\mathbf{c}_j)} = 1 - \left( \frac{d_{(\mathbf{c}_i,\mathbf{c}_j)}}{d_t} \right)^2 \,, \tag{19}$$

where $d_t$ is the threshold for distance. When the distance difference between two correspondences is less than $d_t$, they are considered compatible due to the length consistency of rigid transformations [6]. The hard compatibility matrix can be formulated as:

$$S^h_{(\mathbf{c}_i,\mathbf{c}_j)} = \left\{ \begin{array}{ll} 1\,; & d_{(\mathbf{c}_i,\mathbf{c}_j)} \le d_t \\ 0\,; & d_{(\mathbf{c}_i,\mathbf{c}_j)} > d_t \end{array} \right. \tag{20}$$

However, the first order compatibility measure suffers from outliers due to locality and ambiguity [6]. Following [45], we calculate the second order compatibility scores [6] as edges in the graph. The second order compatibility score between the correspondence pair $(\mathbf{c}_i, \mathbf{c}_j)$ is computed based on the hard compatibility matrix:

$$S^2_{(\mathbf{c}_i,\mathbf{c}_j)} = S^h_{(\mathbf{c}_i,\mathbf{c}_j)} \cdot \sum_{k=1}^{N_c} S^h_{(\mathbf{c}_i,\mathbf{c}_k)} \cdot S^h_{(\mathbf{c}_k,\mathbf{c}_j)} \,, \tag{21}$$

where $N_c$ is the number of input correspondences. Based on the compatibility graph, we select $K$ reliable correspondences as seeds and construct local sets for each seed. Specifically, following [1, 6], we use first-order compatibility scores to compute the leading eigenvectors via the power iteration method [22]. These leading feature vectors serve as confidence scores for reliable seed selection. For each seed, we explore its top-$N_f$ neighbors in the second order measure space. Then, within each neighbor set, we recompute the second-order compatibility score and select the top-$N_2$ ($N_2 < N_f$) correspondences as the local set for the $i$-th seed [6].

## A.2 Evaluation Metrics

**Rotation Error (RE)** measures the geometric distance in degrees between the estimated and ground-truth rotation matrices:

$$\mathrm{RE} = \arccos \left( \frac{\mathrm{trace}\left(\mathbf{R}^T \mathbf{R}_{gt}\right) - 1}{2} \right) \,, \tag{22}$$

where $\mathbf{R}$ denotes the estimated rotation matrix and $\mathbf{R}_{gt}$ denotes the ground-truth rotation matrix.

**Translation Error (TE)** measures the Euclidean distance between the estimated and ground-truth translation vectors:

$$\mathrm{TE} = \|\mathbf{t} - \mathbf{t}_{gt}\|_2 \,, \tag{23}$$

where $\mathbf{t}$ denotes the estimated translation vector and $\mathbf{t}_{gt}$ denotes the ground-truth translation vector.

**Registration Recall (RR)** measures the fraction of correctly registered point cloud pairs whose RE and TE are both below certain thresholds:

$$\text{RR}_{\text{3DMatch\&3DLoMatch}} = \frac{1}{N} \sum_{i=1}^{N} \left[ \text{RE}_i < 15° \wedge \text{TE}_i < 30 \text{ m} \right] .$$

$$\text{RR}_{\text{KITTI}} = \frac{1}{N} \sum_{i=1}^{N} \left[ \text{RE}_i < 5° \wedge \text{TE}_i < 60 \text{ m} \right] .$$

(24)

Following [1, 5, 6, 39], we compute the mean RE and TE only with the correctly registered point cloud pairs .

### A.3 Pseudocode for key parts of our algorithm

There are two key parts in our algorithm: Inlier Identification via $\ell_0$-minimization and two-stage decoupling strategy.

---

**Algorithm 1:** Inlier Identification via $\ell_0$-minimization

---

**Data:** Source point cloud $\mathcal{P} = \left\{ \mathbf{p}_i \in \mathbb{R}^3 \mid i = 1, \ldots, N \right\}$ and target point cloud
$\qquad \mathcal{Q} = \left\{ \mathbf{q}_i \in \mathbb{R}^3 \mid i = 1, \ldots, M \right\}$
**Result:** $\ell_0$-minimization problem of alignment error for each local set

---

1 Establish correspondences $\mathcal{C}$ through feature matching
2 Calculate the compatibility score $S^2_{(\mathbf{c}_i, \mathbf{c}_j)}$ according to Eq. (18)-Eq. (21)
3 Select $N_1$ seed correspondences with high compatibility scores.
4 For $i = 1, 2, \ldots, N_1$
5    Construct $i$-th local set, which containing $N_2$ compatible correspondences.
6    Formulate the $\ell_0$ minimization problem of alignment error for the $i$-th local set according to Eq. (2)
7 EndFor

---

**Algorithm 2:** Two-stage decoupling strategy

---

**Data:** Given point pairs in the $k$-th local set $\mathbf{P}_k = \{\mathbf{p}_{k_i}\}_{i=1}^{K_2}$ and $\mathbf{Q}_k = \{\mathbf{q}_{k_i}\}_{i=1}^{K_2}$, parameters $\lambda_R, \lambda_t, K_R, K_t$

**Result:** Estimated rotation $\mathbf{R}_k^*$ and translation $\mathbf{t}_k^*$ hypothesis for the $k$-th local set

1 % Decoupling the alignment error into a rotation fitting error and a translation fitting error

2 Calculate relative positions $\bar{\mathbf{P}}_k$ and $\bar{\mathbf{Q}}_k$ for all pairs: $\bar{\mathbf{p}}_{k_{ij}} = \mathbf{p}_{k_j} - \mathbf{p}_{k_i}$ and $\bar{\mathbf{q}}_{k_{ij}} = \mathbf{q}_{k_j} - \mathbf{q}_{k_i}$

3 % Decoupling the inlier identification from the rotation estimation

4 Formulate the $\ell_0$-minimization problem for the rotation fitting error $\bar{\mathbf{O}}_k$:

$$\bar{\mathbf{O}}_k^* = \arg\min_{\bar{\mathbf{O}}_k} \|\bar{\mathbf{O}}_k\|_{\ell_0}\,,$$
$$\text{subject to: } \bar{\mathbf{O}}_k = \bar{\mathbf{Q}}_k - \bar{\mathbf{P}}_k \mathbf{R}_k - \bar{\boldsymbol{\Xi}}_k\,. \tag{25}$$

5 % Bayesian-based inlier identification and rotation estimation

6 Construct $\bar{\boldsymbol{\Theta}}_k$ from the left null-space of $\bar{\mathbf{P}}_k$: $\bar{\boldsymbol{\Theta}}_k \bar{\mathbf{P}}_k = \mathbf{0}$

7 Eliminate the components related to $\mathbf{R}_k$ in the constraints of rotation fitting error $\ell_0$-minimization: $\bar{\boldsymbol{\Theta}}_k \bar{\mathbf{O}}_k = \bar{\boldsymbol{\Theta}}_k \bar{\mathbf{Q}}_k - \bar{\boldsymbol{\Theta}}_k \bar{\boldsymbol{\Xi}}_k$

8 Define $\tilde{\bar{\mathbf{Q}}}_k = \bar{\boldsymbol{\Theta}}_k \bar{\mathbf{Q}}_k$ and $\bar{\boldsymbol{\Pi}}_k = \bar{\boldsymbol{\Theta}}_k \bar{\boldsymbol{\Theta}}_k^T$ and formulate the unconstrained problem for rotation fitting error:

$$\min_{\bar{\mathbf{O}}_k} \frac{1}{2} \left\|(\tilde{\bar{\mathbf{Q}}}_k - \bar{\boldsymbol{\Theta}}_k \bar{\mathbf{O}}_k)^T \bar{\boldsymbol{\Pi}}_k^{-1}(\tilde{\bar{\mathbf{Q}}}_k - \bar{\boldsymbol{\Theta}}_k \bar{\mathbf{O}}_k)\right\|_F^2 + \lambda_R \left\|\bar{\mathbf{O}}_k\right\|_F^2\,. \tag{26}$$

9 The explicit solution can be calculated directly:

$$\bar{\mathbf{O}}_k^* = (\bar{\boldsymbol{\Theta}}_k^T \bar{\boldsymbol{\Pi}}_k^{-1} \bar{\boldsymbol{\Theta}}_k + 2\lambda_R \mathbf{I})_k^{-1} \bar{\boldsymbol{\Theta}}_k^T \bar{\boldsymbol{\Pi}}_k^{-1} \tilde{\bar{\mathbf{Q}}}_k\,. \tag{27}$$

10 Solve $\mathbf{R}_k^*$ using SVD on the identified top $K_R$ pairs with the smallest error $\bar{\mathbf{P}}_{\mathcal{I}_R}$ and $\bar{\mathbf{Q}}_{\mathcal{I}_R}$

11 % Decoupling the inlier identification from the translation estimation

12 Based on the estimated $\mathbf{R}_k^*$, the $\ell_0$-minimization problem for the translation fitting error $\hat{\mathbf{O}}_k$ is formulated as:

$$\hat{\mathbf{O}}_k^* = \arg\min_{\hat{\mathbf{O}}_k} \|\hat{\mathbf{O}}_k\|_{\ell_0}\,,$$
$$\text{subject to: } \hat{\mathbf{O}}_k = \mathbf{Q}_k - \mathbf{P}_k \mathbf{R}_k^* - \mathbf{t}_k \mathbf{1}^T - \boldsymbol{\Xi}_k \tag{28}$$

13 Define $\boldsymbol{\Theta}$ satisfying $\boldsymbol{\Theta}\mathbf{1} = \mathbf{0}$. Eliminate the components associated with translation:

$$\boldsymbol{\Theta}_k \hat{\mathbf{O}}_k^T = \boldsymbol{\Theta}_k (\mathbf{Q}_k - \mathbf{P}_k \mathbf{R}_k^*)^T - \boldsymbol{\Theta}_k \boldsymbol{\Xi}_k^T\,. \tag{29}$$

14 % Bayesian-based inlier identification and translation estimation

15 Define $\mathbf{X}_k = \boldsymbol{\Theta}_k (\mathbf{Q}_k^T - (\mathbf{P}_k \mathbf{R}_k^*)^T)$ and $\boldsymbol{\Pi}_k = \boldsymbol{\Theta}_k \boldsymbol{\Theta}_k^T$. Formulate the unconstrained optimization problem:

$$\min_{\hat{\mathbf{O}}_k^T} \frac{1}{2} \left\|(\mathbf{X}_k - \boldsymbol{\Theta}_k \hat{\mathbf{O}}_k^T)^T \boldsymbol{\Pi}_k^{-1}(\mathbf{X}_k - \boldsymbol{\Theta}_k \hat{\mathbf{O}}_k^T)\right\|_F^2 + \lambda_t \left\|\hat{\mathbf{O}}_k^T\right\|_F^2\,. \tag{30}$$

16 The explicit solution can be calculated directly:

$$\hat{\mathbf{O}}_k^* = ((2\lambda_t \mathbf{I} + \boldsymbol{\Theta}_k^T \boldsymbol{\Pi}_k^{-1} \boldsymbol{\Theta}_k)^{-1} \boldsymbol{\Theta}_k^T \boldsymbol{\Pi}_k^{-1} \mathbf{X}_k)^T\,. \tag{31}$$

17 Solve $\mathbf{t}_k^*$ based on the identified top-$K_t$ correspondences with the smallest error $\mathbf{P}_{\mathcal{I}_t}$ and $\mathbf{Q}_{\mathcal{I}_t}$.

---

### A.4 Hyper-parameter selection

The $K_1$ and $K_2$ are set to 30 and 20 for all experiments. For other hyper-parameters ($\lambda_R$, $\lambda_t$, $K_R$ and $K_t$), we employ a grid search strategy with criterion of maximizing inliers. For a given set of

parameters $(\lambda_R, K_R)$, the optimization criterion for $\mathbf{R}_k$ is shown as:

$$\max_{\mathbf{R}_k} \lfloor \mathbb{I}_R \rfloor ,$$
$$\text{subject to: } \bar{\mathbf{q}}_{k_{ij}} - \mathbf{R}_k \bar{\mathbf{p}}_{k_{ij}} = \bar{\xi}_{k_{ij}} , \forall i \in \mathbb{I}_R , \tag{32}$$

where $\mathbb{I}_R$ is the index set of the inlier correspondence pairs and the operation $\lfloor \cdot \rfloor$ denotes the cardinality of the set. Similarly, the selection criterion for the translation vector $\mathbf{t}_k$ is established as follows:

$$\max_{\mathbf{t}_k} \lfloor \mathbb{I}_t \rfloor ,$$
$$\text{subject to: } \mathbf{q}_{k_i} - \mathbf{R}_k^* \mathbf{p}_{k_i} - \mathbf{t}_k = \xi_{k_i} , \forall i \in \mathbb{I}_t , \tag{33}$$

where $\mathbb{I}_R$ is the index set of inlier correspondences.

## A.5   Impact of noise standard deviation

In this section, we visually illustrate the impact of noise standard deviation on the point cloud. As shown in Fig. 7, compared with (a) the clean Bunny model, when the noise standard deviation is increased to 0.01, the geometric structure of the model in (b) remains mostly recognizable. Therefore, 0.01 is often used as the noise standard. However, as the noise standard deviation increases up to 0.1, the geometric structure of the Bunny is severely degraded, going beyond the noise levels typically encountered in robotics and computer vision applications.

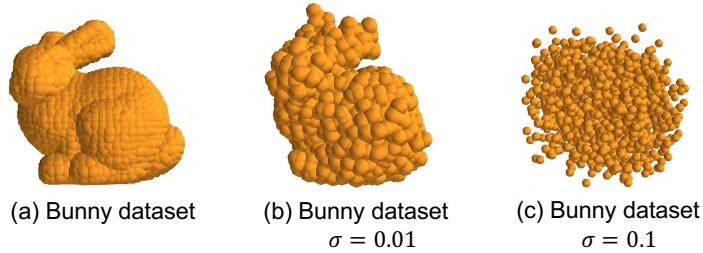

(a) Bunny dataset          (b) Bunny dataset          (c) Bunny dataset
                                  $\sigma = 0.01$                $\sigma = 0.1$

Figure 7: **The impact of Gaussian noise changes on the scanning model.** Bunny point cloud scaled inside unit cube $[0, 1]^3$ and corrupted by different levels of noise and outliers, all viewed from the same perspective angle. (a) Clean Bunny model point cloud. (b) Bunny dataset, generated from (a) by adding isotropic Gaussian noise with a standard deviation $\sigma = 0.01$. (c) Bunny dataset, generated from (a) by adding isotropic Gaussian noise with $\sigma = 0.1$.

## A.6   Efficiency and accuracy.

In this section, we report the inference time, rotation error, and translation error by increasing the corresponding number $N_c$ from 250 to 5000. The curves of FGR, RANSAC, MAC, and our method are flat and difficult to distinguish visually, demonstrating their efficiency. In addition, when there are fewer inputs, the influence of outliers is more obvious. The registration accuracy of FGR and RANSAC decreases significantly as the number of points decreases. The rotation and translation errors of our method are less affected by the number of correspondences. Compared with other methods, our method achieves the most accurate and fast registration for each number of input correspondences.

## A.7   Effectiveness of the two-stage decoupling strategy.

We evaluate the effectiveness of our proposed two-stage decoupling strategy (TDS) by formulating the alignment error $\ell_0$-minimization problem directly on the Bunny data instead of local sets. The rotation and translation are estimated without hypotheses. We provide a comparison with other optimization-based methods [46, 12, 4, 39] as the outlier ratio increases from $0\%$ to $90\%$. Our TDS consistently achieves the highest registration accuracy, demonstrating its inlier identification capability.

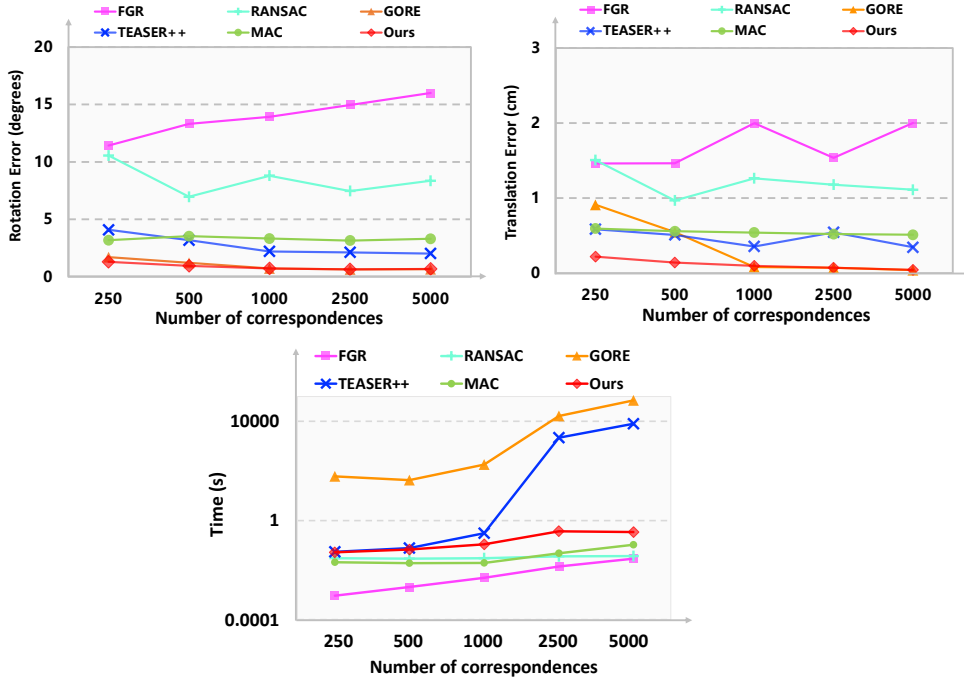

Figure 8: **Comparison results with respect to the number of correspondences.**

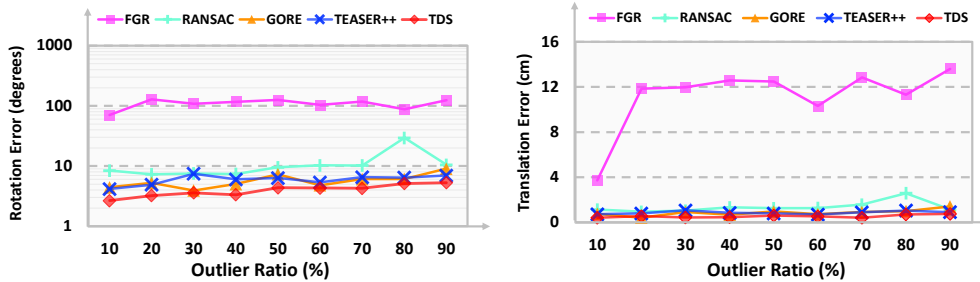

Figure 9: **Comparison results of our two-stage decoupling strategy with optimization-based methods.** We compare the rotation error and translation error of our proposed two-stage decoupling strategy (TDS) with optimization-based methods [46, 12, 4, 39, 45].

### A.8 Additional comparisons.

We also provide a comparison with learning-based registration method EGST [43], we re-evaluate our method under the same dataset settings and metrics as EGST. The comparison results on KITTI and 3DMatch are shown in Table 4 below. The results of EGST reported in the table follow its published paper. Our method shows better performance in rotation error and comparable results in translation error.

Table 4: Comparison results with EGST.

| Method | KITTI | | 3DMatch | |
|---|---|---|---|---|
| | Error(R) | Error(t) | Error(R) | Error(t) |
| EGST | 0.0168 | 0.0018 | 0.2086 | 0.0087 |
| Ours | 0.0059 | 0.0078 | 0.0305 | 0.0059 |

### A.9 Sensitivity to parameters.

We conduct ablation studies on the KITTI dataset to evaluate the sensitivity of our algorithm to various parameters. Firstly, we ablate the number of local sets $N_1$ and correspondences $N_2$ in each local set. As shown in tables below, our method is insensitive to $N_1$ and $N_2$, achieving high registration rates (RR) and low errors (RE and TE). Then, we evaluate the impact of rotation estimation threshold $K_R$ and translation estimation threshold $K_t$. As shown in Fig. 10, the curves of registration metrics (RR, RE and TE) remain stable when $K_R$ and $K_t$ increase, indicating the insensitivity of our method to these parameters.

Table 5: Ablation of the number of local sets.

|    | RR(%)↑ | RE(°)↓ | TE(cm)↓ |
|----|--------|--------|---------|
| 20 | 98.02  | 0.33   | 20.68   |
| 25 | 98.02  | 0.32   | 20.62   |
| 30 | 98.20  | 0.32   | 20.73   |
| 35 | 98.02  | 0.33   | 20.60   |

Table 6: Ablation of the number of correspondences in each local set.

|    | RR(%)↑ | RE(°)↓ | TE(cm)↓ |
|----|--------|--------|---------|
| 5  | 98.02  | 0.33   | 20.72   |
| 10 | 98.02  | 0.33   | 20.69   |
| 15 | 98.02  | 0.33   | 20.68   |
| 20 | 98.20  | 0.32   | 20.73   |

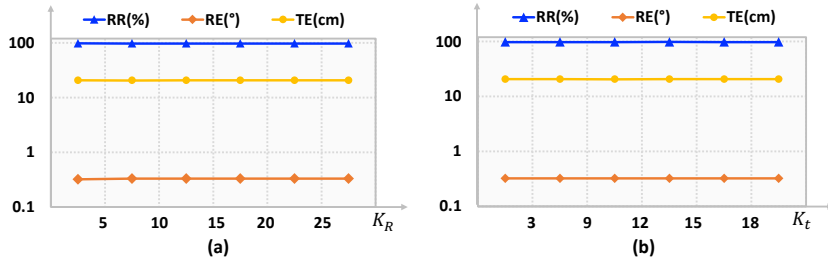

Figure 10: **Ablation of inlier thresholds** $K_R$ **and** $K_t$**.**

The results demonstrate that our method is parameter insensitive, making it reliable in practical implementations.

### A.10 Limitations and broader impact.

We propose a $\ell_0$-norm based method to solve the point cloud registration problem. The method is robust to high outlier ratios and noise, and effective for different numbers of correspondences. It introduces a novel perspective to achieve accurate point cloud registration in practical applications. Our algorithm is most likely to be used in quality inspection and autonomous driving. It can provide fast and accurate alignment between workpieces and templates, as well as enhance localization and scene perception for autonomous vehicles. Furthermore, we wish to test the effectiveness of our method in other areas involving registration tasks, including multimodal medical image registration. One possible situation is a quality inspection scenario, where our algorithm may fail when dealing with large workpiece surfaces without obvious features. Future research will focus on enhancing the robustness of our algorithm to featureless data.

### A.11 Scalability of our algorithm.

Exploring the scalability of our algorithm and its suitability for real-time applications is important for practical deployment. Existing algorithms struggle to achieve both fast speed and high accuracy. Our experiments demonstrate that our algorithm not only achieves high accuracy and robustness but also remains competitive in terms of efficiency, highlighting its potential for real-time applications. The

speed of our method can be further improved through techniques such as parallel computing and C++ implementation. Notably, the two-stage decoupling strategy (TDS) consumes most of the running time (95% of the total), and thus, it could particularly benefit from parallelization. In the first stage of TDS, we compute the relative positions for all point pairs. In the second stage, the computation of null-space matrices also requires substantial processing time. Therefore, these two components are the primary targets for acceleration. Regarding scalability, the proposed two-stage decoupling strategy is a crucial step for inlier identification and can be flexibly combined with other methods to improve accuracy.

## A.12 Qualitative results.

We show qualitative results on 3DMatch [44] and 3DLoMatch [17] in Fig. 12. The yellow and blue point clouds represent the source and target point clouds, respectively. The first column represents the input point clouds and the second column represents the aligned point clouds transformed with the ground-truth transformations. Compared to other methods [6, 45], our approach achieves better alignment results. We also provide registration results on the KITTI [13] dataset in Fig. 13. The input source and target point clouds are in different poses, and the point clouds transformed using our estimated transformations are successfully registered.

The visualization of failure cases is provided in Fig. 11 . We observe that when there are repeated patterns (e.g., similar chairs appearing in different locations) or textureless structures (e.g., walls, floors), failure cases may occur due to the feature matching ambiguity. These remain challenging problems in point cloud registration and have not yet been effectively addressed. Potential solutions include improving feature extraction or applying point cloud completion based on the scene context.

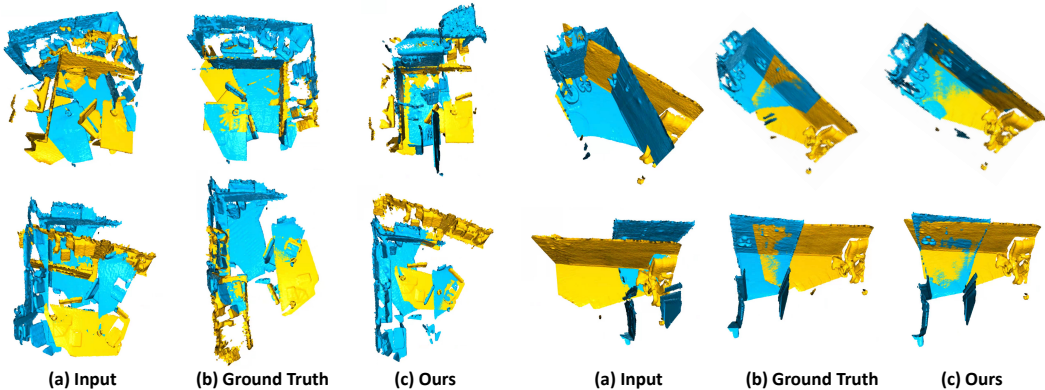

Figure 11: **Failure cases on the 3DMatch dataset.**

## A.13 Datasets.

**Stanford Bunny** The Bunny model from the Stanford 3D Scanning Repository [10] is scanned with a Cyberware 3030 MS scanner, with licensing restrictions against commercial use.Each scan takes the form of a range image, described in the local coordinate system of the scanner. These range images are merged using a modified ICP algorithm [30].

**Odometry KITTI** KITTI [15] is published under the NonCommercial-ShareAlike 3.0 License. It contains 11 sequences scanned by a Velodyne HDL-64 3D laser scanner in outdoor driving scenarios. Following [5, 6], we use sequences 8-10 for testing.

**3DMatch and 3DLoMatch** 3DMatch [44] comprises 62 scenes from SUN3D [36], 7-Scenes [28], RGB-D Scenes v.2 [21], Analysis-by-Synthesis [31], BundleFusion [11], and Halbel et al. [16] (Table. 7). These scenes are captured from diverse indoor environments using different sensors like the Microsoft Kinect and Intel Realsense, and are processed into point cloud fragments by fusing 50 consecutive depth frames using TSDF volumetric fusion [10]. The dataset contains 46 scenes for training, 8 scenes for validation and 8 scenes for testing. The original 3DMatch [44] only considers point cloud pairs with $> 30\%$ overlap. In addition to this benchmark (3DMatch),

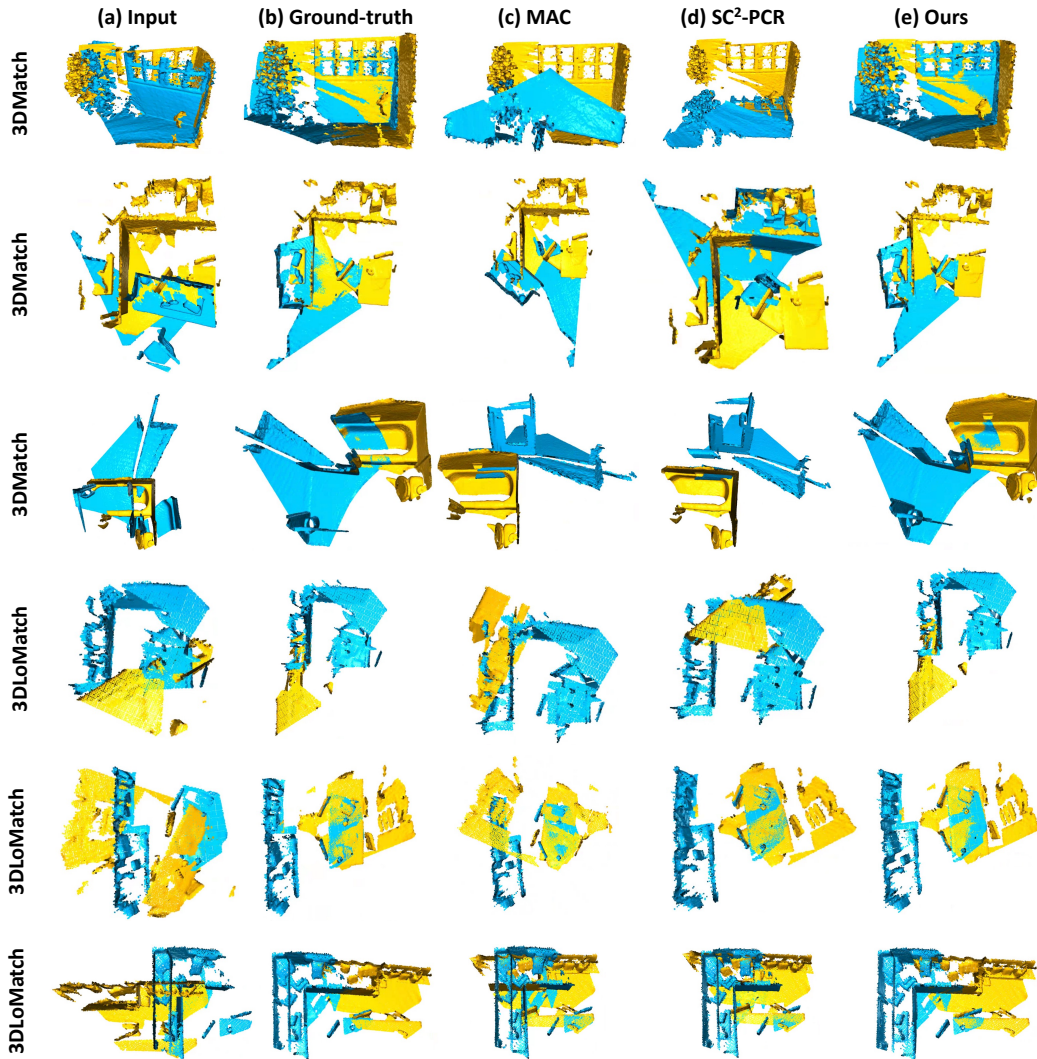

Figure 12: **Qualitative registration results on the 3DMatch and 3DLoMatch datasets.**

we follow [17] to include point cloud pairs with overlaps between $10\%$ and $30\%$ to form another benchmark (3DLoMatch).

Table 7: Raw data used in the 3DMatch dataset and their licenses.

| Datasets | License |
|---|---|
| SUN3D [36] | CC BY-NC-SA 4.0 |
| 7-Scenes [28] | Non-commercial use only |
| RGB-D Scenes v.2 [21] | (License not stated) |
| Analysis-by-Synthesis [31] | CC BY-NC-SA 4.0 |
| BundleFusion [11] | CC BY-NC-SA 4.0 |
| Halbel et al. [16] | CC BY-NC-SA 4.0 |

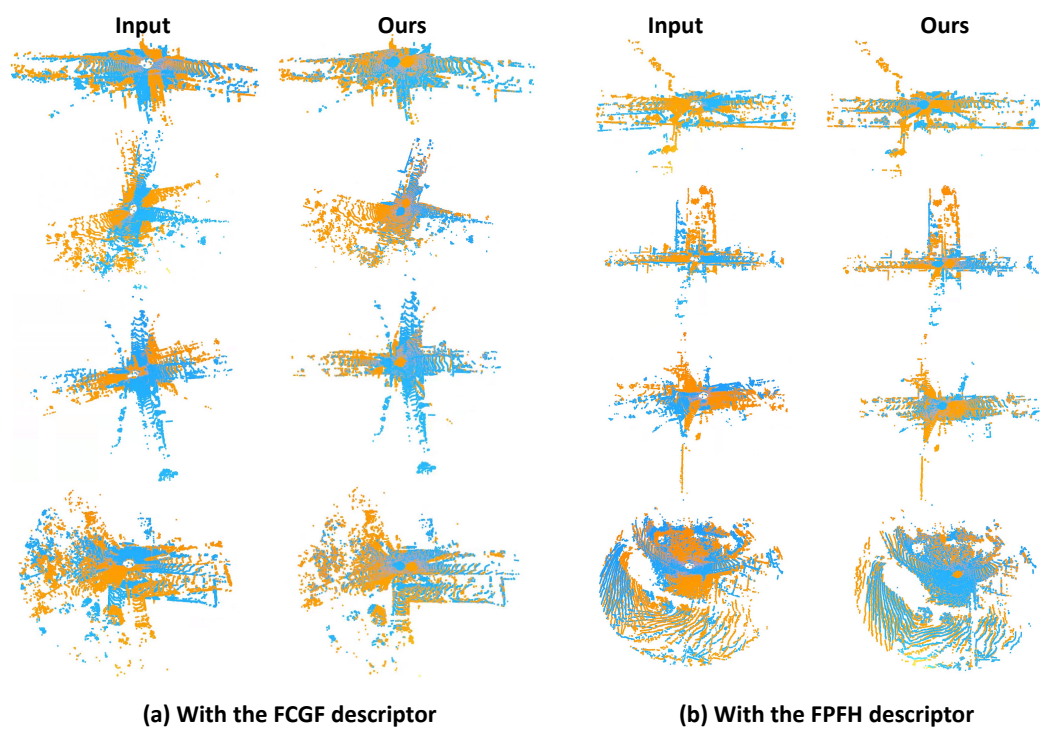

(a) With the FCGF descriptor

(b) With the FPFH descriptor

Figure 13: **Visualizations of registration results on the KITTI dataset.**

